# Probabilistic Low-Rank Subspace Clustering

**S. Derin Babacan**
University of Illinois at Urbana-Champaign
Urbana, IL 61801, USA
dbabacan@gmail.com

**Shinichi Nakajima**
Nikon Corporation
Tokyo, 140-8601, Japan
nakajima.s@nikon.co.jp

**Minh N. Do**
University of Illinois at Urbana-Champaign
Urbana, IL 61801, USA
minhdo@illinois.edu

## Abstract

In this paper, we consider the problem of clustering data points into low-dimensional subspaces in the presence of outliers. We pose the problem using a density estimation formulation with an associated generative model. Based on this probability model, we first develop an iterative expectation-maximization (EM) algorithm and then derive its global solution. In addition, we develop two Bayesian methods based on variational Bayesian (VB) approximation, which are capable of automatic dimensionality selection. While the first method is based on an alternating optimization scheme for all unknowns, the second method makes use of recent results in VB matrix factorization leading to fast and effective estimation. Both methods are extended to handle sparse outliers for robustness and can handle missing values. Experimental results suggest that proposed methods are very effective in subspace clustering and identifying outliers.

## 1 Introduction

Modeling data using low-dimensional representations is a fundamental approach in data analysis, motivated by the inherent redundancy in many datasets and to increase the interpretability of data via dimensionality reduction. A classical approach is principal component analysis (PCA), which implicitly models data to live in a single low-dimensional subspace within the high-dimensional ambient space. However, a more suitable model in many applications is the union of multiple low-dimensional subspaces. This modeling leads to the more challenging problem of *subspace clustering*, which attempts to simultaneously cluster data points into multiple subspaces and find the basis of the corresponding subspace for each cluster.

Mathematically, subspace clustering can be defined as follows: Let $\mathbf{Y}$ be the $M \times N$ data matrix consisting of $N$ vectors $\{\mathbf{y}_i \in \mathbb{R}^M\}_{i=1}^N$, which are assumed be drawn from a union of $K$ linear (or affine) subspaces $\mathcal{S}_k$ of unknown dimensions $d_k = \dim(\mathcal{S}_k)$ with $0 < d_k < M$. The subspace clustering problem is to find the number of subspaces $K$, their dimensions $\{d_k\}_{k=1}^K$, the subspace bases, and the clustering of vectors $\mathbf{y}_i$ into these subspaces.

Subspace clustering is widely investigated problem due to its application in a large number of fields, including computer vision [6, 12, 23], machine learning [11, 22] and system identification [31] (see [22, 28] for comprehensive reviews). Some of the common approaches include algebraic-geometric approaches such as generalized PCA (GPCA) [19, 29], spectral clustering [18], and mixture models [9, 26]. Recently, there has been a great interest in methods based on sparse and/or low-rank representation of the data [5, 7, 8, 14–17, 25]. The general approach in these methods is to first find a sparse/low-rank representation $\mathbf{X}$ of the data and then apply a spectral clustering method on $\mathbf{X}$. It has been shown that with appropriate modeling, $\mathbf{X}$ provides information about the seg-

mentation of the vectors into the subspaces. Two common models for $\mathbf{X}$ are summarized below.

- **Sparse Subspace Clustering (SSC) [7, 25]:** This approach is based on representing data points $\mathbf{y}_i$ as sparse linear combinations of other data points. A possible optimization formulation is

$$\min_{\mathbf{D},\mathbf{X}} \ \beta\|\mathbf{Y} - \mathbf{D}\|_{\mathrm{F}}^2 + \|\mathbf{D} - \mathbf{DX}\|_{\mathrm{F}}^2 + \lambda\|\mathbf{X}\|_1\,, \quad \text{subject to } \mathrm{diag}(\mathbf{X}) = \mathbf{0}\,, \qquad (1)$$

where $\|\cdot\|_{\mathrm{F}}$ is the Frobenius norm and $\|\cdot\|_1$ is the $l_1$-norm.

- **Low-Rank Representation (LRR) [8, 14–17] :** These methods are based on a principle similar to SSC, but $\mathbf{X}$ is modeled as low-rank instead of sparse. A general formulation for this model is

$$\min_{\mathbf{D},\mathbf{X}} \ \beta\|\mathbf{Y} - \mathbf{D}\|_{\mathrm{F}}^2 + \|\mathbf{D} - \mathbf{DX}\|_{\mathrm{F}}^2 + \lambda\|\mathbf{X}\|_*\,, \qquad (2)$$

where $\|\cdot\|_*$ is the nuclear norm.

In these formulations, $\mathbf{D}$ is a clean dictionary and data $\mathbf{Y}$ is assumed to be the noisy version of $\mathbf{D}$ possibly with outliers. When $\beta \to \infty$, $\mathbf{Y} = \mathbf{D}$, and thus the data itself is used as the dictionary [7,15,25]. If the subspaces are disjoint or independent[1], the solution $\mathbf{X}$ in both formulations is shown to be such that $X_{ik} \neq 0$ only if data points $\mathbf{y}_i$ and $\mathbf{y}_k$ belong to the same subspace [7, 14, 15, 25]. That is, the sparsest/lowest rank solution is obtained when each point $\mathbf{y}_i$ is represented as a linear combination of points in its own subspace. The estimated $\mathbf{X}$ is used to define an affinity matrix [18] such as $|\mathbf{X}| + |\mathbf{X}^T|$ and a spectral clustering algorithm, such as normalized cuts [24], is applied on this affinity to cluster the data vectors. The subspace bases can then be obtained in a straightforward manner using this clustering. These methods have also been extended to include sparse outliers.

In this paper, we develop probabilistic modeling and inference procedures based on a principle similarly to LRR. Specifically, we formulate the problem using a latent variable model based on the factorized form $\mathbf{X} = \mathbf{AB}$, and develop inference procedures for estimating $\mathbf{A}$, $\mathbf{B}$, $\mathbf{D}$ (and possibly outliers), along with the associated hyperparameters. We first show a maximum-likelihood formulation of the problem, which is solved using an expectation-maximization (EM) method. We derive and analyze its global solution, and show that it is related to closed-form solution of the rank-minimization formulation (2) in [8]. To incorporate automatic estimation of the latent dimensionality of subspaces and the algorithmic parameters, we further present two Bayesian approaches: The first one is based on same probability model as the EM method, but additional priors are placed on the latent variables and variational Bayesian inference is employed for approximate marginalization to avoid overfitting. The second one is based on a matrix-factorization formulation, and exploits the recent results on Bayesian matrix factorization [20] to achieve fast estimation that is less prone to errors due to alternating optimization. Finally, we extent both methods to handle large errors (outliers) in the data, to achieve robust estimation.

Compared to deterministic methods, proposed Bayesian methods have the advantages of automatically estimating the dimensionality and the algorithmic parameters. This is crucial in unsupervised clustering as the parameters can have a drastic effect on the solution, especially in the presence of heavy noise and outliers. While our methods are closely related to Bayesian PCA [2, 3, 20] and mixture models [9, 26], our formulation is based on a different model and leads to robust estimation less dependent on the initialization, which is one of the main disadvantages of such methods.

## 2 Probabilistic Model for Low-Rank Subspace Clustering

In the following, without loss of generality we assume that $M \leq N$ and $\mathbf{Y}$ is full row-rank. We also assume that each subspace is sufficiently sampled, that is, for each $\mathcal{S}_i$ of dimension $d_i$, there exist at least $d_i$ data vectors $\mathbf{y}_i$ in $\mathbf{Y}$ that span $\mathcal{S}_i$. As for notation, the expectations are denoted by $\langle \cdot \rangle$, $\mathcal{N}$ is the normal distribution, and $\mathrm{diag}()$ denotes the diagonal of a matrix. We do not differentiate the variables from the parameters of the model to have a unified presentation throughout the paper.

We formulate the latent variable model as

$$\mathbf{y}_i = \mathbf{d}_i + \mathbf{n}_Y\,, \qquad (3)$$

$$\mathbf{d}_i = \mathbf{DAb}_i + \mathbf{n}_D\,, \quad i = 1,\ldots,N \qquad (4)$$

where $\mathbf{D}$ is $M \times N$, $\mathbf{A}$ is $N \times N$, and $\mathbf{n}_Y$, $\mathbf{n}_D$ are i.i.d. Gaussian noise independent of the data. The associated probability model is given by[2]

$$p(\mathbf{y}_i|\mathbf{d}_i) = \mathcal{N}\left(\mathbf{y}_i \mid \mathbf{d}_i, \sigma_y^2 \mathbf{I}_M\right), \tag{5}$$

$$p(\mathbf{d}_i|\mathbf{D}, \mathbf{A}, \mathbf{b}_i) = \mathcal{N}\left(\mathbf{d}_i \mid \mathbf{D}\mathbf{A}\mathbf{b}_i, \sigma_d^2 \mathbf{I}_M\right), \tag{6}$$

$$p(\mathbf{b}_i) = \mathcal{N}\left(\mathbf{b}_i \mid \mathbf{0}, \mathbf{I}_N\right). \tag{7}$$

We model the components as independent such that $p(\mathbf{Y}|\mathbf{D}) = \prod_{i=1}^{N} p(\mathbf{y}_i|\mathbf{d}_i)$, $p(\mathbf{D}|\mathbf{A}, \mathbf{B}) = \prod_{i=1}^{N} p(\mathbf{d}_i|\mathbf{D}, \mathbf{A}, \mathbf{b}_i)$, and $p(\mathbf{B}) = \prod_{i=1}^{N} p(\mathbf{b}_i)$. This model has the generative interpretation where latent vectors $\mathbf{b}_i$ are drawn from an isotropic Gaussian distribution, shaped by $\mathbf{A}$ to obtain $\mathbf{A}\mathbf{b}_i$, which then chooses a sample of points from the dictionary $\mathbf{D}$ to generate the $i^{\text{th}}$ dictionary element $\mathbf{d}_i$. In this sense, matrix $\mathbf{D}\mathbf{A}$ has a role similar to principal subspace matrix in probabilistic principal component analysis (PPCA) [26]. However, notice that in contrast to this and related approaches such as mixture of PPCAs [9, 26], the principal subspaces are defined using the data itself in (6).

In (5), the observations $\mathbf{y}_i$ are modeled as corrupted versions of dictionary elements $\mathbf{d}_i$ with iid Gaussian noise. Such separation of $\mathbf{D}$ and $\mathbf{Y}$ is not necessary if there are no outliers, as the presence of noise $\mathbf{n}_Y$ and $\mathbf{n}_D$ makes them unidentifiable. However, we use this general formulation to later include outliers.

## 2.1 An Expectation-Maximization (EM) Algorithm

In (5) - (7), latent variables $\mathbf{b}_i$ can be regarded as missing data and $\mathbf{D}$, $\mathbf{A}$ as parameters, and an EM algorithm can be devised for their joint estimation. The complete log-likelihood is given by

$$\mathcal{L}_{\mathcal{C}} = \sum_{i=1}^{N} \log p(\mathbf{y}_i, \mathbf{b}_i) \tag{8}$$

with $p(\mathbf{y}_i, \mathbf{b}_i) = p(\mathbf{y}_i|\mathbf{d}_i) \, p(\mathbf{d}_i|\mathbf{D}, \mathbf{A}, \mathbf{b}_i) \, p(\mathbf{b}_i)$. The EM algorithm can be obtained by taking the expectation of this log-likelihood with respect to (w.r.t.) $\mathbf{B}$ (E-step) and maximizing it w.r.t. $\mathbf{D}$, $\mathbf{A}$, $\sigma_d$, and $\sigma_y$ (M-step). In the E-step, the distribution $p(\mathbf{B}|\mathbf{D}, \mathbf{A}, \sigma_d^2)$ is found as $\mathcal{N}(\langle \mathbf{B} \rangle, \Sigma_{\mathbf{B}})$ with

$$\langle \mathbf{B} \rangle = \Sigma_{\mathbf{B}} \frac{1}{\sigma_d^2} \mathbf{A}^T \mathbf{D}^T \mathbf{D}, \qquad \Sigma_{\mathbf{B}}^{-1} = \mathbf{I} + \frac{1}{\sigma_d^2} \mathbf{A}^T \mathbf{D}^T \mathbf{D} \mathbf{A}, \tag{9}$$

and the expectation of the likelihood is taken w.r.t. this distribution. In the M-step, maximizing the expected log-likelihood w.r.t. $\mathbf{D}$ and $\mathbf{A}$ in an alternating fashion yields the update equations

$$\mathbf{D} = \frac{1}{\sigma_y^2} \mathbf{Y} \left[ \frac{1}{\sigma_y^2} \mathbf{I} + \frac{1}{\sigma_d^2} \langle (\mathbf{I} - \mathbf{A}\mathbf{B})(\mathbf{I} - \mathbf{A}\mathbf{B})^T \rangle_{\mathbf{B}} \right]^{-1}, \quad \mathbf{A} = \langle \mathbf{B} \rangle^T \left[ \langle \mathbf{B}\mathbf{B}^T \rangle \right]^{-1}, \tag{10}$$

with $\langle \mathbf{B}\mathbf{B}^T \rangle = \mathbf{B}\mathbf{B}^T + N\Sigma_{\mathbf{B}}$. Finally, the estimates of $\sigma_d^2$ and $\sigma_y^2$ are found as

$$\sigma_d^2 = \frac{\|\mathbf{D} - \mathbf{D}\mathbf{A}\langle \mathbf{B} \rangle\|_{\mathrm{F}}^2 + N \operatorname{tr}(\mathbf{A}^T \mathbf{D}^T \mathbf{D} \mathbf{A} \Sigma_{\mathbf{B}})}{MN}, \qquad \sigma_y^2 = \frac{\|\mathbf{Y} - \mathbf{D}\|_{\mathrm{F}}^2}{MN}. \tag{11}$$

In summary, the maximum likelihood solution is obtained by an alternating iterative procedure where first the statistics of $\mathbf{B}$ are calculated using (9), followed by the M-step updates for $\mathbf{D}$, $\mathbf{A}$, $\sigma_d$, and $\sigma_y$ in (10) and (11), respectively.

## 2.2 Global Solution of the EM algorithm

Although the iterative EM algorithm above can be applied to estimate $\mathbf{A}$, $\mathbf{B}$, $\mathbf{D}$, the global solutions can in fact be found in closed form. Specifically, the optimal solution is found (see the supplementary) as either $\mathbf{A}\langle \mathbf{B} \rangle = \mathbf{0}$ or

$$\mathbf{A}\langle \mathbf{B} \rangle = \mathbf{V}_q \left[ \mathbf{I}_q - N\sigma_d^2 \bar{\mathbf{\Lambda}}_q^{-2} \right] \mathbf{V}_q^T, \tag{12}$$

where $\bar{\mathbf{\Lambda}}_q$ is a $q \times q$ diagonal matrix with coefficients $\bar{\lambda}_j = \max(\lambda_j, \sqrt{N}\sigma_d)$. Here, $\mathbf{D} = \mathbf{U}\mathbf{\Lambda}\mathbf{V}^T$ is the singular value decomposition (SVD) of $\mathbf{D}$, and $\mathbf{V}_q$ contains its $q$ right singular vectors that correspond to singular values that are larger than or equal to $\sqrt{N}\sigma_d$. Hence, the solution (12) is related to the rank-$q$ shape interaction matrix (SIM) $\mathbf{V}_q\mathbf{V}_q^T$ [6], while in addition it involves scaling of the singular vectors via thresholded singular values of $\mathbf{D}$.

Using $\mathbf{A}\langle\mathbf{B}\rangle$ in (10), the singular vectors of the optimal $\mathbf{D}$ and $\mathbf{Y}$ are found to be the same, and the singular values $\lambda_j$ of $\mathbf{D}$ are related to the singular values $\xi_j$ of $\mathbf{Y}$ as

$$\xi_j = \begin{cases} \lambda_j + N\sigma_y^2\,\lambda_j^{-1}, & \text{if} \quad \lambda_j > \sqrt{N}\sigma_d \\ \lambda_j \frac{\sigma_y^2 + \sigma_d^2}{\sigma_d^2}, & \text{if} \quad \lambda_j \leq \sqrt{N}\sigma_d \end{cases} \tag{13}$$

This is a combination of two operations: down-scaling and the solutions a quadratic equation, where the latter is a polynomial thresholding operation on the singular values $\xi_j$ of $\mathbf{Y}$ (see supplementary). Hence, the optimal $\mathbf{D}$ is obtained by applying the thresholding operation (13) on the singular values of $\mathbf{Y}$, where the shrinkage amount is small for large singular values so that they are preserved, whereas small singular values are shrank by down-scaling. This is an interesting result, as there is no explicit penalty on the rank of $\mathbf{D}$ in our modeling. As shown in [8], the nuclear norm formulation (2) leads to a similar closed-form solution, but it requires the solution of a quartic equation.

Finally, at the stationary points, the noise variance $\sigma_d^2$ is found as

$$\sigma_d^2 = \frac{1}{N-q} \sum_{q'=q+1}^{N} \lambda_{q'}^2, \tag{14}$$

that is, the average of the squared discarded singular values of $\mathbf{D}$ when computing $\mathbf{D}\mathbf{A}\langle\mathbf{B}\rangle$. A simple closed form expression of $\sigma_y^2$ cannot be found due to the polynomial thresholding in (13), but it can simply be calculated using (11).

In summary, if $\sigma_y^2$ and $\sigma_d^2$ are given, the optimal $\mathbf{D}$ and $\mathbf{A}\langle\mathbf{B}\rangle$ are found by taking the SVD of $\mathbf{Y}$ and applying shrinkage/thresholding operations on the singular values of $\mathbf{Y}$. However, this method requires setting $\sigma_y^2$ and $\sigma_d^2$ manually. When $\mathbf{Y}$ itself is used as the dictionary $\mathbf{D}$ (i.e., $\sigma_y^2 = 0$), an alternative method is to choose $q$, the total number of independent dimensions to be retained in $\mathbf{D}\mathbf{A}\langle\mathbf{B}\rangle$, calculate $\sigma_d^2$ from (14), and finally use (12) to obtain $\mathbf{A}\langle\mathbf{B}\rangle$. However, when $\sigma_y^2 \neq 0$, $q$ cannot directly be set and a trial-and-error procedure is required to find it. Although $\sigma_d^2$ and $\sigma_y^2$ can also be estimated automatically using the iterative EM procedure in Sec. 2.1, this method is susceptible to local minima, as the trivial solution $\mathbf{A}\langle\mathbf{B}\rangle = \mathbf{0}$ also maximizes the likelihood.

These issues can be overcome by employing a Bayesian estimation to automatically determine the effective dimensionality of $\mathbf{D}$ and $\mathbf{AB}$. We develop two methods towards this goal, which are described next.

## 3 Variational Bayesian Low-Rank Subspace Clustering

Bayesian estimation of $\mathbf{D}$, $\mathbf{A}$ and $\mathbf{B}$ can be achieved by treating them as latent variables to be marginalized over to avoid overfitting and trivial solutions such as $\mathbf{AB} = \mathbf{0}$. Here we develop such a method based on the probability model in the previous section but with additional priors introduced on $\mathbf{A}$, $\mathbf{B}$ and the noise variances. Before presenting our complete probability model, we first introduce the matrix-variate normal distribution as its use significantly simplifies the algorithm derivation. For a $M \times N$ matrix $\mathbf{X}$, the matrix-variate normal distribution is given by [10]

$$\mathcal{N}(\mathbf{X}|\mathbf{M}, \mathbf{\Sigma}, \mathbf{\Omega}) = (2\pi)^{\frac{NM}{2}} |\mathbf{\Sigma}|^{-\frac{N}{2}} |\mathbf{\Omega}|^{-\frac{M}{2}} \exp\left[-\frac{1}{2}\operatorname{tr}\left(\mathbf{\Sigma}^{-1}\left(\mathbf{X} - \mathbf{M}\right)\mathbf{\Omega}^{-1}\left(\mathbf{X} - \mathbf{M}\right)^T\right)\right] \tag{15}$$

where $\mathbf{M}$ is the mean, and $\mathbf{\Sigma}$, $\mathbf{\Omega}$ are $M \times M$ row and $N \times N$ column covariances, respectively.

To automatically determine the number of principal components in $\mathbf{AB}$, we employ an automatic relevance determination mechanism [21] on the columns of $\mathbf{A}$ and rows of $\mathbf{B}$ using priors $\mathrm{p}(\mathbf{A}) = \mathcal{N}(\mathbf{A}|\mathbf{0}, \mathbf{I}, \mathbf{C_A})$, $\mathrm{p}(\mathbf{B}) = \mathcal{N}(\mathbf{B}|\mathbf{0}, \mathbf{C_B}, \mathbf{I})$, where $\mathbf{C_A}$ and $\mathbf{C_B}$ are diagonal matrices with $\mathbf{C_A} = \operatorname{diag}(c_{A,i})$ and $\mathbf{C_B} = \operatorname{diag}(c_{B,i})$, $i = 1, \ldots, N$. Jeffrey's priors are placed on $c_{A,i}$ and $c_{B,i}$, and they are assumed to be independent. To avoid scale ambiguity, the columns of $\mathbf{A}$ and rows of $\mathbf{B}$ can also be coupled using the same set of hyperparameters $\mathbf{C_A} = \mathbf{C_B}$, as in [1].

For inference, we employ the variational Bayesian (VB) method [4] which leads to a fast algorithm. Let $q(\mathbf{D}, \mathbf{A}, \mathbf{B}, \mathbf{C_A}, \mathbf{C_B}, \sigma_d^2, \sigma_y^2)$ be the distribution that approximates the posterior. The variational free energy is given by the following functional

$$\mathcal{F} = \langle \log q(\mathbf{D}, \mathbf{A}, \mathbf{B}, \mathbf{C_A}, \mathbf{C_B}, \sigma_d^2, \sigma_y^2) - \log p(\mathbf{Y}, \mathbf{D}, \mathbf{A}, \mathbf{B}, \mathbf{C_A}, \mathbf{C_B}, \sigma_d^2, \sigma_y^2) \rangle . \tag{16}$$

Using the mean field approximation, the approximate posterior is factorized as $q(\mathbf{D}, \mathbf{A}, \mathbf{B}, \mathbf{C_A}, \mathbf{C_B}, \sigma_d^2, \sigma_y^2) = q(\mathbf{D}) q(\mathbf{A}) q(\mathbf{B}) q(\mathbf{C_A}) q(\mathbf{C_B}) q(\sigma_d^2) q(\sigma_y^2)$. Using the priors defined above with the conditional distributions in (5) and (6), the approximating distributions of $\mathbf{D}$, $\mathbf{A}$ and $\mathbf{B}$ minimizing the free energy $\mathcal{F}$ are found as matrix-variate normal distributions[3] $q(\mathbf{D}) = \mathcal{N}(\langle \mathbf{D} \rangle, \mathbf{I}, \mathbf{\Omega_D})$, $q(\mathbf{A}) = \mathcal{N}(\langle \mathbf{A} \rangle, \mathbf{\Sigma_A}, \mathbf{\Omega_A})$ and $q(\mathbf{B}) = \mathcal{N}(\langle \mathbf{B} \rangle, \mathbf{\Sigma_B}, \mathbf{I})$, with parameters

$$\langle \mathbf{D} \rangle = \frac{1}{\langle \sigma_y^2 \rangle} \mathbf{Y} \, \mathbf{\Omega_D}, \qquad \mathbf{\Omega_D^{-1}} = \left( \frac{1}{\langle \sigma_y^2 \rangle} \right) \mathbf{I}_N + \frac{1}{\langle \sigma_d^2 \rangle} \langle (\mathbf{I} - \mathbf{AB})(\mathbf{I} - \mathbf{AB})^T \rangle \tag{17}$$

$$\mathbf{\Sigma_A^{-1}} = \frac{1}{N} \operatorname{tr}(\mathbf{C_A^{-1}} \mathbf{\Omega_A}) \, \mathbf{I} + \frac{1}{N \sigma_d^2} \operatorname{tr}(\mathbf{\Omega_A} \langle \mathbf{BB}^T \rangle) \langle \mathbf{D}^T \mathbf{D} \rangle \tag{18}$$

$$\mathbf{\Omega_A^{-1}} = \frac{1}{N} \operatorname{tr}(\mathbf{\Sigma_A}) \mathbf{C_A^{-1}} + \frac{1}{N \sigma_d^2} \operatorname{tr}(\mathbf{\Sigma_A} \langle \mathbf{D}^T \mathbf{D} \rangle) \langle \mathbf{BB}^T \rangle \tag{19}$$

$$\langle \mathbf{A} \rangle \mathbf{C_A^{-1}} + \frac{1}{\sigma_d^2} \langle \mathbf{D}^T \mathbf{D} \rangle \langle \mathbf{A} \rangle \langle \mathbf{BB}^T \rangle = \frac{1}{\sigma_d^2} \langle \mathbf{D}^T \mathbf{D} \rangle \langle \mathbf{B} \rangle^T \tag{20}$$

$$\langle \mathbf{B} \rangle = \mathbf{\Sigma_B} \frac{1}{\langle \sigma_d^2 \rangle} \langle \mathbf{A}^T \mathbf{D}^T \mathbf{D} \rangle, \qquad \mathbf{\Sigma_B^{-1}} = \mathbf{C_B^{-1}} + \frac{1}{\langle \sigma_d^2 \rangle} \langle \mathbf{A}^T \mathbf{D}^T \mathbf{D} \mathbf{A} \rangle . \tag{21}$$

The estimate $\langle \mathbf{A} \rangle$ in (20) is solved using fixed-point iterations. The hyperparameter updates are given by

$$\langle c_{A,i}^{-1} \rangle = \frac{N}{\langle \mathbf{A}^T \mathbf{A} \rangle_{ii}}, \qquad \langle c_{B,i}^{-1} \rangle = \frac{N}{\operatorname{diag}(\langle \mathbf{BB}^T \rangle_{ii})}, \tag{22}$$

$$\langle \sigma_d^2 \rangle = \frac{\langle \| \mathbf{D} - \mathbf{DAB} \|_{\mathrm{F}}^2 \rangle}{MN}, \qquad \langle \sigma_y^2 \rangle = \frac{\langle \| \mathbf{Y} - \mathbf{D} \|_{\mathrm{F}}^2 \rangle}{MN} . \tag{23}$$

Explicit forms of the required moments are given in the supplementary. In summary, the algorithm alternates between calculating the sufficient statistics of the distributions of $\mathbf{D}$, $\mathbf{A}$ and $\mathbf{B}$, and the updates of the hyperparameters $c_{A,i}$, $c_{B,i}$, $\sigma_d^2$ and $\sigma_y^2$. The convergence can be monitored during iterations using the variational free energy $\mathcal{F}$. $\mathcal{F}$ is also useful in model comparison, which we use for detecting outliers, as explained in Sec. 5.

Similarly to the matrix factorization approaches [2, 3, 13], automatic dimensionality selection is invoked via hyperparameters $c_{A,i}$ and $c_{B,i}$, which enforce sparsity in the columns and rows of $\mathbf{A}$ and $\mathbf{B}$, respectively. Specifically, when a particular set of variances $c_{A,i}$, $c_{B,i}$ assume very small values, the posteriors of the $i^{\mathrm{th}}$ column of $\mathbf{A}$ and $i^{\mathrm{th}}$ row of $\mathbf{B}$ will be concentrated around zero, such that the effective number of principal directions in $\mathbf{AB}$ will be reduced. In practice, this is performed via thresholding of variances $c_{A,i}$, $c_{B,i}$ with a small threshold (e.g., $10^{-10}$).

## 4  A Factorization-Based Variational Bayesian Approach

Another Bayesian method can be developed by further investigating the probability model. Essentially, the estimates of $\mathbf{A}$ and $\mathbf{B}$ is based on the factorization of $\mathbf{D}$ and are independent of $\mathbf{Y}$. Thus, one can apply a matrix factorization method to $\mathbf{D}$, and relate this factorization to $\mathbf{DAB}$ to find $\mathbf{AB}$. Based on this idea, we modify the probabilistic model to $p(\mathbf{D}) = \mathcal{N}(\mathbf{D} | \mathbf{D}_L \mathbf{D}_R, \mathbf{I}, \frac{1}{\sigma_d^2} \mathbf{I})$, $p(\mathbf{D}_L) = \mathcal{N}(\mathbf{D}_L | \mathbf{0}, \mathbf{I}, \mathbf{C}_L)$, $p(\mathbf{D}_R) = \mathcal{N}(\mathbf{D}_R | \mathbf{0}, \mathbf{C}_R, \mathbf{I})$, where diagonal covariances $\mathbf{C}_L$ and $\mathbf{C}_R$ are used to induce sparsity in the columns of $\mathbf{D}_L$ and rows of $\mathbf{D}_R$, respectively. It has been shown in [20] that when variational Bayesian inference is applied to this model, the global solution is found analytically and given by

$$\mathbf{D}_L \mathbf{D}_R = \mathbf{U} \mathbf{\Lambda}_F \mathbf{V}^T , \tag{24}$$

where $\mathbf{U}$, $\mathbf{V}$ contain the singular vectors of $\mathbf{D}$, and $\mathbf{\Lambda}_F$ is a diagonal matrix, obtained by applying a specific shrinkage method to the singular values of $\mathbf{D}$ [20]. The number of retained singular values are therefore automatically determined. Then, setting $\mathbf{D}_L\mathbf{D}_R$ equal to $\mathbf{DAB}$, we obtain the solution $\mathbf{AB} = \mathbf{V}_f\mathbf{\Lambda}_f^{-1}\mathbf{\Lambda}_F\mathbf{V}_f^T$, where the subscript $f$ denotes the retained singular value and vectors.

The only modification to the method in the previous section is to replace the estimation of $\mathbf{A}$ and $\mathbf{B}$ in (18)-(21) with the global solution $\mathbf{V}_f\mathbf{\Lambda}_f^{-1}\mathbf{\Lambda}_F\mathbf{V}_f^T$. Thus, this method allows us to avoid the alternating optimization for finding $\mathbf{A}$ and $\mathbf{B}$, which potentially can get stuck in undesired local minima. Although the probability model is slightly different than the one described in the previous section, we anticipate that its global solution to be related to the factorization-based solution.

## 5  Robustness to Outliers

Depending on the application, the outliers might be in various forms. For instance in motion tracking applications, an entire data point might become an outlier if the tracker fails at that instance. In other applications, only a subset of coordinates might be corrupted with large errors. Both types (and possibly others) can be handled in our modeling. The only required change in the model is in the conditional distribution of the observations as

$$p(\mathbf{Y}|\mathbf{D}) = \mathcal{N}(\mathbf{Y}|\mathbf{D} + \mathbf{E}, \sigma_y^2), \tag{25}$$

where $\mathbf{E}$ is the sparse outlier matrix for which we introduce the prior

$$p(\mathbf{E}) = \mathcal{N}(\mathbf{E}|\mathbf{0}, \mathbf{C}_\mathbf{E}^C, \mathbf{C}_\mathbf{E}^R) = \mathcal{N}(\text{vec}(\mathbf{E})|\mathbf{0}, \mathbf{C}_\mathbf{E}^C \otimes \mathbf{C}_\mathbf{E}^R). \tag{26}$$

The shape of the column covariance matrix $\mathbf{C}_\mathbf{E}^C$ and row covariance matrix $\mathbf{C}_\mathbf{E}^R$ depends on the nature of outliers. If only entire data points might be corrupted, we can use $\mathbf{C}_\mathbf{E}^C = \mathbf{I}$ and independent terms in $\mathbf{C}_\mathbf{E}^R$ such that $\mathbf{C}_\mathbf{E}^R = \text{diag}(c_{E,i}^R)$, $i = 1, \dots, N$. When entire coordinates can be corrupted, row-sparsity in $\mathbf{E}$ can be imposed using $\mathbf{C}_\mathbf{E}^R = \mathbf{I}$ and $\mathbf{C}_\mathbf{E}^C = \text{diag}(c_{E,i}^C)$. In the first case, the VB estimation rule becomes $q(\mathbf{e}_i) = \mathcal{N}(\langle \mathbf{e}_i \rangle, \mathbf{I}, \mathbf{\Sigma}_{\mathbf{e}_i})$ with

$$\langle \mathbf{e}_i \rangle = \mathbf{\Sigma}_{\mathbf{e}_i} \frac{1}{\langle \sigma_y^2 \rangle} \ (\mathbf{y}_i - \langle \mathbf{d}_i \rangle) \quad \mathbf{\Sigma}_{\mathbf{e}_i} = \text{diag}\left( \frac{1}{\langle \sigma_y^2 \rangle} + \frac{1}{\langle c_{E,i}^R \rangle} \right)^{-1}, \tag{27}$$

with the hyperparameter update $\langle c_{E,i}^R \rangle = \langle \mathbf{e}_i \rangle^T \langle \mathbf{e}_i \rangle + \text{tr}\left( \mathbf{\Sigma}_{\mathbf{e}_i} \right)$. The estimation rules for other outlier models can be derived in a similar manner.

In the presence of outlier data points, there is an inherent unidentifiability between $\mathbf{AB}$ and $\mathbf{E}$ which can prevent the detection of outliers and hence reduce the performance of subspace clustering. Specifically, an outlier $\mathbf{y}_i$ can be included in the sparse component as $\mathbf{e}_i = \mathbf{y}_i$ or included in the dictionary $\mathbf{D}$ with its own subspace, which leads to $(\mathbf{AB})_{ii} \approx 1$. To avoid the latter case, we introduce a heuristic inspired by the birth and death method in [9]. During iterations, data points $\mathbf{y}_i$ with $(\mathbf{AB})_{ii}$ larger than a threshold (e.g., 0.95) are assigned to the sparse component $\mathbf{e}_i$. As this might initially increase the variational energy $\mathcal{F}$, we monitor its progress over a few iterations and reject this "birth" of the sparse component if $\mathcal{F}$ does not decrease below its original state. This method is observed to be very effective in identifying outliers and alleviating the effect of the initialization.

Finally, missing values in $\mathbf{Y}$ can also be handled by modifying the distribution of the observations in (5) to $p(\mathbf{y}_i|\mathbf{d}_i) = \prod_{k \in Z_i} \mathcal{N}\left( y_{ik} \,|\, d_{ik}, \sigma_y^2 \right)$, where $Z_i$ is the set containing the indices of the observed entries in vector $\mathbf{y}_i$. The inference procedures can be modified with relative ease to accommodate this change.

## 6  Experiments

In this section, we evaluate the performance of the three algorithms introduced above, namely, the EM method in Sec. 2.2, the variational Bayesian method in Sec. 3 (VBLR) and the factorization-based method in Sec. 4 (VBLR-Fac). We also include comparisons with deterministic subspace clustering and mixture of PPCA (MPPCA) methods. In all experiments, the estimated $\mathbf{AB}$ matrix is used to find the affinity matrix and the normalized cuts algorithm [24] is applied to find the clustering and hence the subspaces.

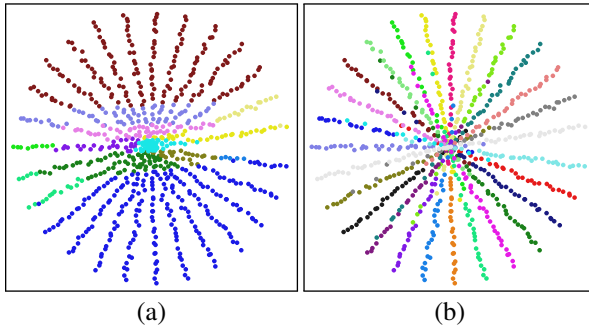

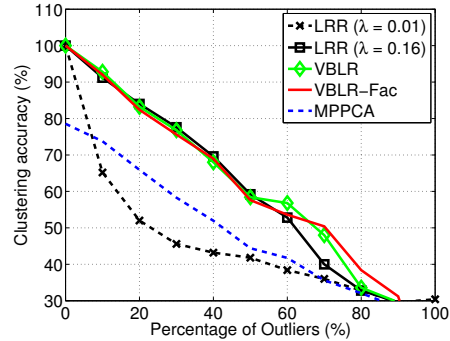

|       | (a) | (b) |
|-------|-----|-----|

Figure 1: Clustering 1D subspaces (points in the same cluster are in the same color) (a) MPPCA [3] result, (b) the result of the EM algorithm (global solution). The Bayesian methods give results almost identical to (b).

Figure 2: Accuracy of clustering 5 independent subspaces of dimension 5 for different percentage of outliers.

**Synthetic Data.** We generated 27 line segments intersecting at the origin, as shown in Fig. 1, where each contains 800 points slightly corrupted by iid Gaussian noise of variance $0.1$. Each line can be considered as a separate 1D subspace, and the subspaces are disjoint but not independent. We first applied the mixture of PPCA [3] to which we provided the dimensions and the number of the subspaces. This method is sensitive to the proximity of the subspaces, and in all of our trials gave results similar to Fig. 1(a), where close lines are clustered together. On the other hand, the EM method accurately clusters the lines into different subspaces (Fig. 1(b)), and it is extremely efficient involving only one SVD. Both Bayesian methods VBLR and VBLR-Fac gave similar results and accurately estimated the subspace dimensions, while the VB-variant of MPPCA [9] gave results similar to Fig. 1(a).

Next, similarly to the setup in [15], we construct 5 independent subspaces $\{\mathcal{S}_i\} \subset \mathbb{R}^{50}$ of dimension 5 with bases $\mathbf{U}_i$ generated as follows: We first generate a random $50 \times 5$ orthogonal matrix $\mathbf{U}_1$, and then rotate it with random orthonormal matrices $\mathbf{R}_i$ to obtain $\mathbf{U}_i = \mathbf{R}_i\mathbf{U}_1$, $i = 2, 3, 4$. Dictionary $\mathbf{D}$ is obtained by sampling 25 points from each subspace using $\mathbf{D}_i = \mathbf{U}_i\mathbf{V}_i$ where $\mathbf{V}_i$ are $5 \times 25$ matrices with elements drawn from $\mathcal{N}(0, 1)$. Finally, $\mathbf{Y}$ is obtained by corrupting $\mathbf{D}$ with outliers sampled from $\mathcal{N}(0, 1)$ and normalized to lie on the unit sphere. We applied our methods VBLR and VBLR-Fac to cluster the data into 5 groups, and compare their performance with MPPCA and LRR. Average clustering errors (over 20 trials) in Fig. 2 show that LRR and the proposed methods provide much better performance than MPPCA. VBLR and VBLR-Fac gave similar results, while VBLR-Fac converges much faster (generally about 10 vs 100 iterations). Although LRR also gives very good results, its performance varies with its parameters. As an example, we included its results obtained by the optimal and a slightly different parameter value, where in the latter case the degradation in accuracy is evident.

Table 1: Clustering errors (%) on the Hopkins155 motion database

| Method | GPCA [19] | LSA [30] | SSC [7] | LRR [15] | VBLR | VBLR-Fac |
|--------|-----------|----------|---------|----------|------|----------|
| Mean   | 30.51     | 8.77     | 3.66    | 1.71     | 1.75 | 1.85     |
| Max    | 55.67     | 38.37    | 37.44   | 32.50    | 35.13| 37.32    |
| Std    | 11.79     | 9.80     | 7.21    | 4.85     | 4.92 | 5.10     |

**Real Data with Small Corruptions.** The Hopkins155 motion database [27] is frequently used to test subspace clustering methods. It consists of 156 sequences where each contains 39 to 550 data vectors corresponding to either 2 or 3 motions. Each motion corresponds to a subspace and each sequence is regarded as a separate clustering task. While most existing methods use a pre-processing stage that generally involves dimensionality reduction using PCA, we do not employ pre-processing and apply our Bayesian methods directly (the EM method cannot handle outliers and thus is not included in the experiments). The mean and maximum clustering errors and the standard deviation in the whole set are shown in Table 1. The proposed methods provide close to state-of-the-art performance, while competing methods require manual tuning of their parameters, which can affect their performance. For instance, the results of LRR is obtained by setting its parameter $\lambda = 4$, while changing it to $\lambda = 2.4$ gives $3.13\%$ error [15]. The method in [8], which is similar to our EM-

method except that it also handles outliers, achieves an error rate of $1.44\%$. Finally, the deterministic method [17] achieves an error rate of $0.85\%$ and to our knowledge, is the best performing method in this dataset.

**Real Data with Large Corruptions.** To test our methods in real data with large corruptions, we use the Extended Yale Database B [12] where we chose the first 10 classes that contain 640 frontal face images. Each class contains 64 images and each image is resized to $48 \times 42$ and stacked to generate the data vectors. Figure 3 depicts some example images, where significant corruption due to shadows and heavy noise is evident. The task is to cluster the 640 images into 10 classes. The segmentation accuracies achieved by the proposed methods and some existing methods are listed in Table 2, where it is evident that the proposed methods achieve state-of-art-performance. Example recovered clean dictionary and sparse outlier components are shown in Fig. 3.

Table 2: Clustering accuracy ($\%$) on the Extended Yale Database B

| Method | LSA [30] | SSC [7] | LRR [15] | VBLR | VBLR-Fac |
|---------|----------|---------|----------|-------|----------|
| Average | 31.72 | 37.66 | 62.53 | 69.72 | 67.62 |

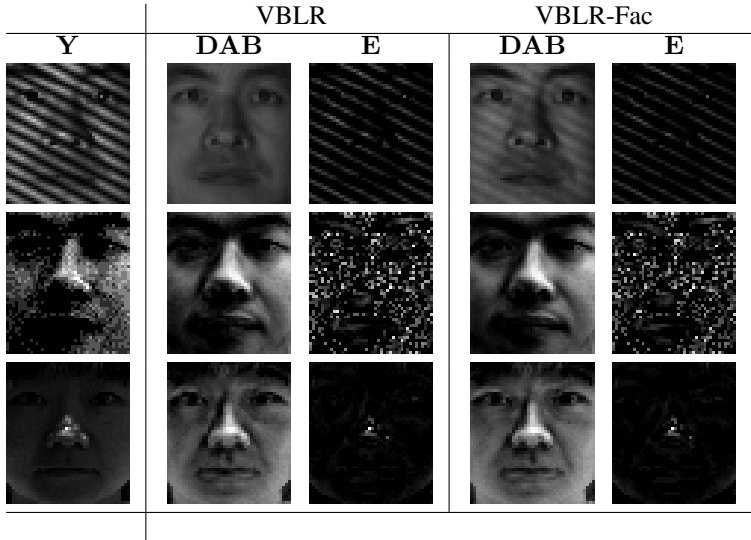

Figure 3: Examples of recovered clean data and large corruptions. Original images are shown in the left column (denoted by $\mathbf{Y}$), the clean dictionary elements obtained by VBLR and VBLR-Fac are shown in columns denoted by $\mathbf{DAB}$, and columns denoted by $\mathbf{E}$ show corruption captured by the sparse element.

## 7 Conclusion

In this work we presented a probabilistic treatment of low dimensional subspace clustering. Using a latent variable formulation, we developed an expectation-maximization method and derived its global solution. We further proposed two effective Bayesian methods both based on the automatic relevance determination principle and variational Bayesian approximation for inference. While the first one, VBLR, relies completely on alternating optimization, the second one, VBLR-Fac, makes use of the global solution of VB matrix factorization to eliminate one alternating step and leads to faster convergence. Both methods have been extended to handle sparse large corruptions in the data for robustness. These methods are advantageous over deterministic methods as they are able to automatically determine the total number of principal dimensions and all required algorithmic parameters. This property is particularly important in unsupervised settings. Finally, our formulation can potentially be extended for modeling multiple nonlinear manifolds, by the use of kernel methods.

**Acknowledgments.** The authors thank anonymous reviewers for helpful comments. SDB acknowledges the Beckman Institute Postdoctoral Fellowship. SN thanks the support from MEXT Kakenhi 23120004. MND was partially supported by NSF CHE 09-57849.

## Footnotes

[1]The subspaces $\mathcal{S}_k$ are called *independent* if $\dim(\bigoplus_{k=1}^K \mathcal{S}_K) = \sum_{k=1}^K \dim(\mathcal{S}_k)$ with $\bigoplus$ the direct sum. The subspaces are *disjoint* if they only intersect at the origin.

[2]Here we assume that $\mathbf{A}\mathbf{b}_i \neq \mathbf{w}_i$ where $\mathbf{w}_i$ is a zero vector with 1 as the $i^{\text{th}}$ coefficient, to have a proper density. This is a reasonable assumption if each subspace is sufficiently sampled and the dictionary element $\mathbf{d}_i$ belongs to one of them (i.e., it is not an outlier). Outliers are explicitly handled later.

[3]The optimal distribution $q(\mathbf{A})$ does not have a matrix-variate normal form. However, we force it to have this form for computational efficiency (see supplementary for details).

# References

[1] S. D. Babacan, M. Luessi, R. Molina, and A. K. Katsaggelos. Sparse Bayesian methods for low-rank matrix estimation. *IEEE Trans. Signal Proc.*, 60(8), Aug 2012.

[2] C. M. Bishop. Bayesian principal components. In *NIPS*, volume 11, pages 382–388, 1999.

[3] C. M. Bishop. Variational principal components. In *Proc. of ICANN*, volume 1, pages 514–509, 1999.

[4] C.M. Bishop. *Pattern Recognition and Machine Learning*. Springer, 2006.

[5] E. J. Candès, X. Li, Y. Ma, and J. Wright. Robust principal component analysis? *CoRR*, abs/0912.3599, 2009.

[6] J. P. Costeira and T. Kanade. A multibody factorization method for independently moving objects. *Int. J. Comput. Vision*, 29(3):159–179, September 1998.

[7] E. Elhamifar and R. Vidal. Sparse subspace clustering. In *CVPR*, pages 2790–2797, 2009.

[8] P. Favaro, R. Vidal, and A. Ravichandran. A closed form solution to robust subspace estimation and clustering. In *CVPR*, pages 1801–1807, 2011.

[9] Z. Ghahramani and M. J. Beal. Variational inference for Bayesian mixtures of factor analysers. In *NIPS*, volume 12, pages 449–455, 2000.

[10] A. K. Gupta and D. K. Nagar. *Matrix Variate Distributions*. Chapman & Hall/CRC, New York, 2000.

[11] K. Huang and S. Aviyente. Sparse representation for signal classification. In *NIPS*, 2006.

[12] K.-C. Lee, J. Ho, and D. Kriegman. Acquiring linear subspaces for face recognition under variable lighting. *IEEE Trans. Pattern Anal. Machine Intell.*, 27:684–698, 2005.

[13] Y. J. Lim and T. W. Teh. Variational Bayesian approach to movie rating prediction. In *Proc. of KDD Cup and Workshop*, 2007.

[14] G. Liu, Z. Lin, S. Yan, J. Sun, Y. Yu, and Y. Ma. Robust recovery of subspace structures by low-rank representation. *CoRR*, abs/1010.2955, 2012.

[15] G. Liu, Z. Lin, and Y. Yu. Robust subspace segmentation by low-rank representation. In *ICML*, pages 663–670, 2010.

[16] G. Liu, H. Xu, and S. Yan. Exact subspace segmentation and outlier detection by low-rank representation. In *AISTATS*, 2012.

[17] G. Liu and S. Yan. Latent low-rank representation for subspace segmentation and feature extraction. In *ICCV*, 2011.

[18] U. Luxburg. A tutorial on spectral clustering. *Statistics and Computing*, 17(4):395–416, December 2007.

[19] Y. Ma, A. Yang, H. Derksen, and R. Fossum. Estimation of subspace arrangements with applications in modeling and segmenting mixed data,. *SIAM Review*, 50(3):413–458, 2008.

[20] S. Nakajima and M. Sugiyama. Theoretical analysis of Bayesian matrix factorization. *Journal of Machine Learning Research*, 12:2583–2648, 2011.

[21] R. M. Neal. *Bayesian Learning for Neural Networks*. Springer, 1996.

[22] H. Peterkriegel, P. Kroger, and A. Zimek. Clustering high-dimensional data: a survey on subspace slustering, pattern-based clustering, and correlation clustering. In *Proc. KDD*, 2008.

[23] S. Rao, R. Tron, R. Vidal, and Y. Ma. Motion segmentation in the presence of outlying, incomplete, or corrupted trajectories. *IEEE Trans. Pattern Anal. Machine Intell.*, 32(10):1832–1845, 2010.

[24] J. Shi and J. Malik. Normalized cuts and image segmentation. *IEEE Trans. Pattern Anal. Machine Intell.*, 22(8):888 –905, aug 2000.

[25] M. Soltanolkotabi and E. J. Candès. A geometric analysis of subspace clustering with outliers. *CoRR*, 2011.

[26] M. E. Tipping and C. M. Bishop. Mixtures of probabilistic principal component analyzers. *Neural Comput.*, 11(2):443–482, February 1999.

[27] R. Tron and R. Vidal. A benchmark for the comparison of 3-d motion segmentation algorithms. In *CVPR*, June 2007.

[28] R. Vidal. Subspace clustering. *IEEE Signal Process. Mag.*, 28(2):52–68, 2011.

[29] R. Vidal, Y. Ma, and S. Sastry. Generalized principal component analysis (gpca). *IEEE Trans. on PAMI*, 27(12):1945–1959, 2005.

[30] J. Yan and M. Pollefeys. A general framework for motion segmentation: Independent, articulated, rigid, non-rigid, degenerate and non-degenerate. In *ECCV*, volume 4, pages 94–106, 2006.

[31] C. Zhang and R. R. Bitmead. Subspace system identification for training-based MIMO channel estimation. *Automatica*, 41:1623–1632, 2005.

